# Extracting and Learning an Unknown Grammar with Recurrent Neural Networks

**C.L.Giles\*, C.B. Miller**
NEC Research Institute
4 Independence Way
Princeton, N.J. 08540
giles@research.nj.nec.com

**D. Chen, G.Z. Sun, H.H. Chen, Y.C. Lee**
\*Institute for Advanced Computer Studies
Dept. of Physics and Astronomy
University of Maryland
College Park, Md 20742

## Abstract

Simple second-order recurrent networks are shown to readily learn small *known* regular grammars when trained with positive and negative strings examples. We show that similar methods are appropriate for learning *unknown* grammars from examples of their strings. The training algorithm is an *incremental* real-time, recurrent learning (RTRL) method that computes the complete gradient and updates the weights at the end of each string. After or *during* training, a dynamic clustering algorithm extracts the production rules that the neural network has learned. The methods are illustrated by *extracting* rules from *unknown* deterministic regular grammars. For many cases the *extracted* grammar outperforms the neural net from which it was extracted in correctly classifying unseen strings.

## 1 INTRODUCTION

For many reasons, there has been a long interest in "language" models of neural networks; see [Elman 1991] for an excellent discussion. The orientation of this work is somewhat different. The focus here is on what are good measures of the computational capabilities of recurrent neural networks. Since currently there is little theoretical knowledge, what problems would be "good" experimental benchmarks? For discrete inputs, a natural choice would be the problem of learning formal grammars - a "hard" problem even for regular grammars [Angluin, Smith 1982]. Strings of grammars can be presented one character at a time and strings can be of arbitrary length. However, the strings themselves would be, for the most part, feature independent. Thus, the learning capabilities would be, for the most part, feature independent and, therefore insensitive to feature extraction choice.

The learning of *known* grammars by recurrent neural networks has shown promise, for example [Cleeresman, et al 1989], [Giles, et al 1990, 1991, 1992], [Pollack 1991], [Sun, et al 1990], [Watrous, Kuhn 1992a,b], [Williams, Zipser 1988]. But what about learning *unknown* grammars? We demonstrate in this paper that not only can *unknown* grammars be learned, but it is possible to *extract* the grammar from the neural network, both during and after training. Furthermore, the extraction process requires no a priori knowledge about the

grammar, except that the grammar's representation can be *regular*, which is always true for a grammar of bounded string length; which is the grammatical "training sample."

## 2   FORMAL GRAMMARS

We give a brief introduction to grammars; for a more detailed explanation see [Hopcroft & Ullman, 1979]. We define a grammar as a 4-tuple (**N, V, P, S**) where **N** and **V** are nonterminal and terminal vocabularies, **P** is a finite set of production rules and **S** is the start symbol. All grammars we discuss are deterministic and regular. For every grammar there exists a language - the set of strings the grammar generates - and an automaton - the machine that recognizes (classifies) the grammar's strings. For regular grammars, the recognizing machine is a deterministic finite automaton (DFA). There exists a one-to-one mapping between a DFA and its grammar. Once the DFA is known, the production rules are the ordered triples *(node, arc, node)*.

Grammatical inference [Fu 1982] is defined as the problem of finding (learning) a grammar from a finite set of strings, often called the training sample. One can interpret this problem as devising an inference engine that learns and extracts the grammar, see Figure 1.

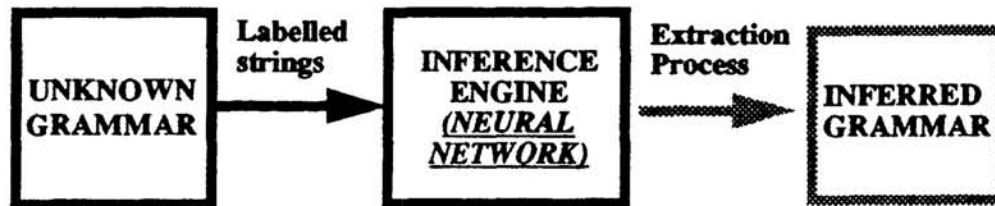

Figure 1: Grammatical inference

For a training sample of positive and negative strings and no knowledge of the unknown regular grammar, the problem is NP-complete (for a summary, see [Angluin, Smith 1982]). It is possible to construct an inference engine that consists of a recurrent neural network and a rule extraction process that yields an inferred grammar.

## 3   RECURRENT NEURAL NETWORK

### 3.1   ARCHITECTURE

Our recurrent neural network is quite simple and can be considered as a simplified version of the model by [Elman 1991]. For an excellent discussion of recurrent networks full of references that we don't have room for here, see [Hertz, et al 1991].

A fairly general expression for a recurrent network (which has the same computational power as a DFA) is:

$$S_i^{t+1} = F(S_j^t, I^t; W)$$

where $F$ is a nonlinearity that maps the state neuron $S^t$ and the input neuron $I^t$ at time $t$ to the next state $S^{t+1}$ at time $t+1$. The weight matrix $W$ parameterizes the mapping and is usually learned (however, it can be totally or partially programmed). A DFA has an analogous mapping but does not use $W$. For a recurrent neural network we define the mapping $F$ and order of the mapping in the following manner [Lee, et al 1986]. For a **first-order** recurrent net:

where N is the number of hidden state neurons and L the number of input neurons; $W_{ij}$ and $Y_{ij}$ are the real-valued weights for respectively the state and input neurons; and $\sigma$ is a stan-

$$S_i^{t+1} = \sigma \left( \sum_j^N W_{ij} S_j^t + \sum_k^L Y_{ik} I_k^t \right)$$

dard sigmoid discriminant function. The values of the hidden state neurons $S^t$ are defined in the finite N-dimensional space $[0,1]^N$. Assuming all weights are connected and the net is fully recurrent, the weight space complexity is bounded by $O(N^2+NL)$. *Note that the input and state neurons are not the same neurons.* This representation has the capability, assuming sufficiently large N and L, to represent any state machine. Note that there are non-trainable unit weights on the recurrent feedback connections.

The natural **second-order** extension of this recurrent net is:

$$S_i^{t+1} = \sigma \left( \sum_{j,k}^{N,L} W_{ijk} S_j^t S_k^t \right) \Rightarrow \sigma \left( \sum_{j,k}^{N,L} W_{ijk} S_j^t I_k^t \right)$$

where certain state neurons become input neurons. Note that the weights $W_{ijk}$ modify a product of the hidden $S_j$ and input $I_k$ neurons. This quadratic form directly represents the state transition diagrams of a state automata process -- *(input, state)* $\Rightarrow$ *(next-state)* and thus makes the state transition mapping very easy to learn. It also permits the net to be *directly programmed* to be a particular DFA. Unpublished experiments comparing first and second order recurrent nets confirm this ease-in-learning hypothesis. The space complexity (number of weights) is $O(LN^2)$. For L«N, both first- and second-order are of the same complexity, $O(N^2)$.

## 3.2   SUPERVISED TRAINING & ERROR FUNCTION

The error function is defined by a special recurrent *output neuron* which is checked at the end of each string presentation to see if it is on or off. By convention this *output neuron* should be on if the string is a positive example of the grammar and off if negative. In practice an error tolerance decides the on and off criteria; see [Giles, et al 1991] for detail. [If a multiclass recognition is desired, another error scheme using many output neurons can be constructed.] We define two error cases: (1) the network fails to reject a negative string (the output neuron is on); (2) the network fails to accept a positive string (the output neuron is off). This accept or reject occurs at the *end of each string* - we define this problem as *inference* versus *prediction*. There is no prediction of the next character in the string sequence. As such, *inference* is a more difficult problem than *prediction*. If knowledge of the classification of every substring of every string exists and alphabetical training order is preserved, then the *prediction* and *inference* problems are equivalent.

The training method is real-time recurrent training (RTRL). For more details see [Williams, Zipser 1988]. The error function is defined as:

$$E = (1/2)\, (Target - S_o^f)^2$$

where $S_o^f$ is the output neuron value at the final time step $t=f$ when the final character is presented and *Target* is the desired value of (1,0) for (positive, negative) examples. Using gradient descent training, the weight update rule for a second-order recurrent net becomes:

$$W_{lmn} = -\alpha \nabla E = \alpha\,(Target - S_o^f) \cdot \frac{\partial S_o^f}{\partial W_{lmn}}$$

where $\alpha$ is the learning rate. From the recursive network state equation we obtain the relationship between the derivatives of $S^t$ and $S^{t+1}$:

$$\frac{\partial S_i^t}{\partial W_{lmn}} = \sigma' \cdot \left[ \delta_{il} S_m^{t-1} I_n^{t-1} + \sum_{jk}^{NL} W_{ijk} I_k^{t-1} \frac{\partial S_j^{t-1}}{\partial W_{lmn}} \right]$$

where $\sigma'$ is the derivative of the discriminant function. This permits on-line learning with partial derivatives calculated iteratively at each time step. Let $\partial S^{t=0}/\partial W_{lmn} = 0$. Note that the space complexity is $O(L^2N^4)$ which can be prohibitive for large N and full connectivity. It is important to note that for all training discussed here, the full gradient is calculated as given above.

## 3.3  PRESENTATION OF TRAINING SAMPLES

The training data consists of a series of stimulus-response pairs, where the stimulus is a string of 0's and 1's, and the response is either "1" for positive examples or "0" for negative examples. The positive and negative strings are generated by an *unknown* source grammar (created by a program that creates random grammars) prior to training. At each discrete time step, *one symbol* from the string activates *one input neuron*, the other input neurons are zero (one-hot encoding). *Training is on-line and occurs after each string presentation; there is no total error accumulation as in batch learning;* contrast this to the batch method of [Watrous, Kuhn 1992]. An extra *end symbol* is added to the string alphabet to give the network more power in deciding the best final neuron state configuration. This requires another input neuron and does not increase the complexity of the DFA (only $N^2$ more weights). The sequence of strings presented during training is very important and certainly gives a bias in learning. We have performed many experiments that indicate that training with alphabetical order with an equal distribution of positive and negative examples is much faster and converges more often than random order presentation.

The *training algorithm is on-line, incremental.* A small portion of the training set is preselected and presented to the network. The net is trained at the end of each string presentation. Once the net has learned this small set or reaches a maximum number of epochs (set before training, 1000 for experiments reported), a small number of strings (10) classified incorrectly are chosen from the rest of the training set and added to the pre-selected set. This small string increment prevents the training procedure from driving the network too far towards any local minima that the misclassified strings may represent. Another cycle of epoch training begins with the augmented training set. If the net correctly classifies all the training data, the net is said to *converge.* The total number of cycles that the network is permitted to run is also limited, usually to about 20.

## 4  RULE EXTRACTION (DFA GENERATION)

As the network is training (or after training), we apply a procedure we call d*ynamic state partitioning (dsp)* for extracting *the network's current conception of the DFA it is learning or has learned.* The rule extraction process has the following steps: 1) clustering of DFA states, 2) constructing a transition diagram by connecting these states together with the alphabet-labelled transitions, 3) putting these transitions together to make the full digraph - forming cycles, and 4) reducing the digraph to a minimal representation. The hypothesis is that during training, the network begins to partition (or quantize) its state space into fairly well-separated, distinct regions or clusters, which represent corresponding states in some DFA. See [Cleeremans, et al 1989] and [Watrous and Kuhn 1992a] for other clustering methods. A simple way of finding these clusters is to divide each neuron's range [0,1] into $q$ partitions of equal size. For N state neurons, $qN$ partitions. For example, for $q=2$, the values of $S^t \geq 0.5$ are 1 and $S^t < 0.5$ are 0, and there are 2N regions with $2^N$ possible values. Thus for N hidden neurons, there exist $q^N$ possible *regions*. The DFA is constructed by generating

a state transition diagram -- associating an input symbol with a set of hidden neuron *partitions* that it is currently in and the set of neuron *partitions* it activates. This ordered triple is also a **production rule**.The initial *partition*, or start state of the DFA, is determined from the initial value of $S^{t=0}$. If the next input symbol maps to the same *partition* we assume a loop in the DFA. Otherwise, a new state in the DFA is formed.This constructed DFA may contain a maximum of $q^N$ states; in practice it is usually much less, since not all neuron partition sets are ever reached. This is basically a tree pruning method and different DFA could be generated based on the choice of branching order. The *extracted* DFA can then be reduced to its minimal size using standard minimization algorithms (an $O(N^2)$ algorithm where N is the number of DFA states) [Hopcroft, Ullman 1979]. [This minimization procedure does not change the grammar of the DFA; the unminimized DFA has same time complexity as the minimized DFA. The process just rids the DFA of redundant, unnecessary states and reduces the space complexity.] *Once the DFA is known, the production rules are easily extracted.*

Since many partition values of q are available, *many DFA can be extracted.* How is the q that gives the best DFA chosen? Or viewed in another way, using different q, what DFA gives the best representation of the grammar of the training set? One approach is to use different q's (starting with q=2), different branching order, different runs with different numbers of neurons and different initial conditions, and see if any similar sets of DFA emerge. Choose the DFA whose similarity set has the smallest number of states and appears most often - an Occam's razor assumption. Define the guess of the DFA as $DFA_g$.This method seems to work fairly well. Another is to see which of the DFA give the best performance on the training set, assuming that the training set is not perfectly learned. We have little experience with this method since we usually train to perfection on the training set. *It should be noted that this DFA extraction method may be applied to any discrete-time recurrent net, regardless of network order or number of hidden layers.* Preliminary results on first-order recurrent networks show that the same DFA are extracted as second-order, but the first-order nets are less likely to converge and take longer to converge than second-order.

## 5   SIMULATIONS - GRAMMARS LEARNED

Many different small (< 15 states) regular *known* grammars have been learned successfully with both first-order [Cleeremans, et al 1989] and second-order recurrent models [Giles, et al 91] and [Watrous, Kuhn 1992a]. In addition [Giles, et al 1990 & 1991] and [Watrous, Kuhn 1992b] show how corresponding DFA and production rules can be extracted. *However for all of the above work, the grammars to be learned were already known.* What is more interesting is the learning of *unknown grammars.*

In figure 2b is a randomly generated minimal 10-state regular grammar created by a program in which the only inputs are the number of states of the unminimized DFA and the alphabet size *p*. (A good estimate of the number of possible unique DFA is ($n2^n n^{pn}/n!$) [Alon, et al 1991] where n is number of DFA states) The shaded state is the start state, filled and dashed arcs represent 1 and 0 transitions and all final states have a shaded outer circle. This *unknown* (honestly, we didn't look) DFA was learned with both 6 and 10 hidden state neuron second-order recurrent nets using the first 1000 strings in alphabetical training order (we could ask the *unknown* grammar for strings). Of two runs for both 10 and 6 neurons, both of the 10 and one of the 6 converged in less than 1000 epochs. (The initial weights were all randomly chosen between [1,-1] and the learning rate and momentum were both 0.5.) Figure 2a shows one of the unminimized DFA that was extracted for a partition parameter of q=2. The minimized 10-state DFA, figure 3b, appeared for q=2 for one 10 neuron net and for q=2,3,4 of the converged 6 neuron net. Consequently, using our previous criteria, we chose this DFA as $DFA_g$, our guess at the unknown grammar. We then asked

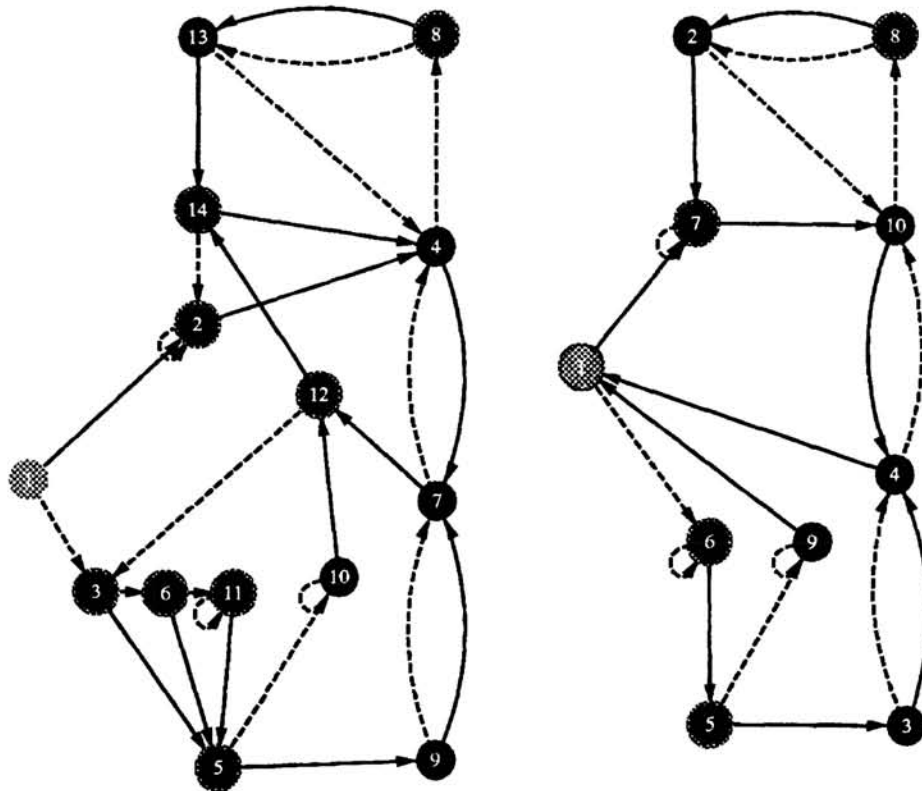

Figures 2a & 2b. Unminimized and minimized 10-state random grammar.

the program what the grammar was and discovered we were correct in our guess. The other minimized DFA for different q's were all unique and usually very large (number of states > 100).

The *trained* recurrent nets were then checked for generalization errors on all strings up to length 15. All made a small number of errors, usually less than 1% of the total of 65,535 strings. However, the correct extracted DFA was perfect and, of course, makes no errors on strings of any length. *Again, [Giles, et al 1991, 1992], the extracted DFA outperforms the trained neural net from which the DFA was extracted.*

Figures 3a and 3b, we see the dynamics of DFA extraction as a 4 hidden neuron neural network is learning as a function of epoch and partition size. This is for grammar Tomita-4 [Giles, et al 1991, 1992]] - a 4-state grammar that rejects any string which has more than three 0's in a row. The number of states of the extracted DFA starts out small, then increases, and finally decreases to a constant value as the grammar is learned. As the partition q of the neuron space increases, the number of minimized and unminimized states increases. When the grammar is learned, the number of minimized states becomes constant and, as expected, the number of minimized states, independent of q, becomes the number of states in the grammar's DFA - 4.

## 6   CONCLUSIONS

Simple recurrent neural networks are capable of learning small regular *unknown* grammars rather easily and generalize fairly well on unseen grammatical strings. The training results are fairly independent of the initial values of the weights and numbers of neurons. For a well-trained neural net, the generalization performance on long unseen strings can be perfect.

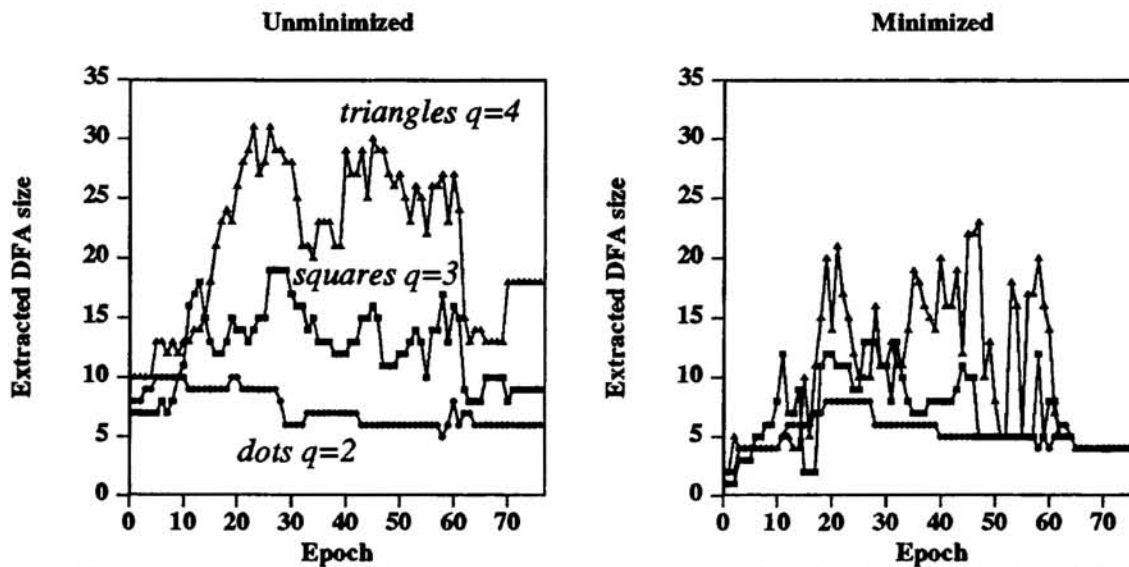

Figures 3a & 3b. Size of number of states (unminimized and minimized) of DFA versus training epoch for different partition parameter q. The correct state size is 4.

A heuristic algorithm called dynamic state partitioning was created to *extract* deterministic finite state automata (DFA) from the neural network, both *during* and *after* training. Using a standard DFA minimization algorithm, the extracted DFA can be reduced to an *equivalent* minimal-state DFA which has reduced space (not time) complexity. When the source or generating grammar is unknown, a good guess of the unknown grammar $DFA_g$ can be obtained from the minimal DFA that is most often extracted from different runs with different numbers of neurons and initial conditions. *From the extracted DFA, minimal or not, the production rules of the learned grammar are evident.*

There are some interesting aspects of the extracted DFA. Each of the unminimized DFA seems to be unique, even those with the same number of states. For recurrent nets that converge, it is often possible to *extract* DFA that are perfect, i.e. the grammar of the unknown source grammar. For these cases all unminimized DFA whose minimal sizes have the same number of states constitute a large *equivalence class* of neural-net-generated DFA, and have the *same performance* on string classification. This *equivalence class* extends across neural networks which vary both in size (number of neurons) and initial conditions. Thus, the *extracted DFA* gives a good indication of how well the neural network learns the grammar.

**In fact, for most of the trained neural nets, the extracted $DFA_g$ outperforms the trained neural networks in classification of unseen strings.** (By definition, a perfect DFA will correctly classify all unseen strings). This is not surprising due to the possibility of error accumulation as the neural network classifies *long* unseen strings [Pollack 1991]. However, when the neural network has learned the grammar well, its generalization performance can be perfect on all strings tested [Giles, et al 1991, 1992]. Thus, the neural network can be considered as a tool for extracting a DFA that is representative of the unknown grammar. Once the $DFA_g$ is obtained, it can be used independently of the trained neural network.

The learning of small DFA using second-order techniques and the full gradient computation reported here and elsewhere [Giles, et al 1991, 1992], [Watrous, Kuhn 1992a, 1992b] give a strong impetus to using these techniques for learning DFA. The question of DFA state capacity and scalability is unresolved. Further work must show how well these ap-

proaches can model grammars with large numbers of states and establish a theoretical and experimental relationship between DFA state capacity and neural net size.

**Acknowledgments**

The authors acknowledge useful and helpful discussions with E. Baum, M. Goudreau, G. Kuhn, K. Lang, L. Valiant, and R. Watrous. The University of Maryland authors gratefully acknowledge partial support from AFOSR and DARPA.

**References**

N. Alon, A.K. Dewdney, and T.J.Ott, 'Efficient Simulation of Finite Automata by Neural Nets, *Journal of the ACM*, Vol 38, p. 495 (1991).

D. Angluin, C.H. Smith, Inductive Inference: Theory and Methods, *ACM Computing Surveys*, Vol 15, No 3, p. 237, (1983).

A. Cleeremans, D. Servan-Schreiber, J. McClelland, Finite State Automata and Simple Recurrent Recurrent Networks, *Neural Computation*, Vol 1, No 3, p. 372 (1989).

J.L. Elman, Distributed Representations, Simple Recurrent Networks, and Grammatical Structure, *Machine Learning*, Vol 7, No 2/3, p. 91 (1991).

K.S. Fu, *Syntactic Pattern Recognition and Applications*, Prentice-Hall, Englewood Cliffs, N.J. Ch.10 (1982).

C.L. Giles, G.Z. Sun, H.H. Chen, Y.C. Lee, D. Chen, Higher Order Recurrent Networks & Grammatical Inference, *Advances in Neural Information Systems 2*, D.S. Touretzky (ed), Morgan Kaufmann, San Mateo, Ca, p.380 (1990).

C.L. Giles, D. Chen, C.B. Miller, H.H. Chen, G.Z. Sun, Y.C. Lee, Grammatical Inference Using Second-Order Recurrent Neural Networks, *Proceedings of the International Joint Conference on Neural Networks*, IEEE91CH3049-4, Vol 2, p.357 (1991).

C.L. Giles, C.B. Miller, D. Chen, H.H. Chen, G.Z. Sun, Y.C. Lee, Learning and Extracting Finite State Automata with Second-Order Recurrent Neural Networks, *Neural Computation*, accepted for publication (1992).

J. Hertz, A. Krogh, R.G. Palmer, *Introduction to the Theory of Neural Computation*, Addison-Wesley, Redwood City, Ca., Ch. 7 (1991).

J.E. Hopcroft, J.D. Ullman, *Introduction to Automata Theory, Languages, and Computation*, Addison Wesley, Reading, Ma. (1979).

Y.C. Lee, G. Doolen, H.H. Chen, G.Z. Sun, T. Maxwell, H.Y. Lee, C.L. Giles, Machine Learning Using a Higher Order Correlational Network, *Physica D*, Vol 22-D, No1-3, p. 276 (1986).

J.B. Pollack, The Induction of Dynamical Recognizers, *Machine Learning*, Vol 7, No 2/3, p. 227 (1991).

G.Z. Sun, H.H. Chen, C.L. Giles, Y.C. Lee, D. Chen, Connectionist Pushdown Automata that Learn Context-Free Grammars, *Proceedings of the International Joint Conference on Neural*, Washington D.C., Lawrence Erlbaum Pub., Vol I, p. 577 (1990).

R.L. Watrous, G.M. Kuhn, Induction of Finite-State Languages Using Second-Order Recurrent Networks, *Neural Computation*, accepted for publication (1992a) and *these proceedings*, (1992b).

R.J. Williams, D. Zipser, A Learning Algorithm for Continually Running Fully Recurrent Neural Networks, *Neural Computation*, Vol 1, No 2, p. 270, (1989).